# Blind separation of delayed and convolved sources.

**Te-Won Lee**
Max-Planck-Society, GERMANY,
AND Interactive Systems Group
Carnegie Mellon University
Pittsburgh, PA 15213, USA
tewon@cs.cmu.edu

**Anthony J. Bell**
Computational Neurobiology,
The Salk Institute
10010 N. Torrey Pines Road
La Jolla, California 92037, USA
tony@salk.edu

**Russell H. Lambert**
Dept of Electrical Engineering
University of South California, USA
rlambert@sipi.usc.edu

## Abstract

We address the difficult problem of separating multiple speakers with multiple microphones in a real room. We combine the work of Torkkola and Amari, Cichocki and Yang, to give Natural Gradient information maximisation rules for recurrent (IIR) networks, blindly adjusting delays, separating and deconvolving mixed signals. While they work well on simulated data, these rules fail in real rooms which usually involve *non-minimum phase* transfer functions, not-invertible using stable IIR filters. An approach that sidesteps this problem is to perform infomax on a feedforward architecture in the frequency domain (Lambert 1996). We demonstrate real-room separation of two natural signals using this approach.

## 1 The problem.

In the linear blind signal processing problem ([3, 2] and references therein), $N$ signals, $\mathbf{s}(t) = [s_1(t) \dots s_N(t)]^T$, are transmitted through a medium so that an array of $N$ sensors picks up a set of signals $\mathbf{x}(t) = [x_1(t) \dots x_N(t)]^T$, each of which

has been mixed, delayed and filtered as follows:

$$x_i(t) = \sum_{j=1}^{N} \sum_{k=0}^{M-1} a_{ijk} s_j(t - D_{ij} - k) \tag{1}$$

(Here $D_{ij}$ are entries in a matrix of delays and there is an $M$-point filter, $\mathbf{a}_{ij}$, between the the $j$th source and the $i$th sensor.) The problem is to invert this mixing without knowledge of it, thus recovering the original signals, $\mathbf{s}(t)$.

## 2 Architectures.

The obvious architecture for inverting eq.1 is the *feedforward* one:

$$u_i(t) = \sum_{j=1}^{N} \sum_{k=0}^{M-1} w_{ijk} x_j(t - d_{ij} - k). \tag{2}$$

which has filters, $\mathbf{w}_{ij}$, and delays, $d_{ij}$, which supposedly reproduce, at the $u_i$, the original uncorrupted source signals, $s_i$. This was the architecture implicitly assumed in [2]. However, it cannot solve the delay-compensation problem, since in eq.1 each delay, $D_{ij}$, delays a single source, while in eq.2 each delay, $d_{ij}$ is associated with a mixture, $x_j$.

Torkkola [8], has addressed the problem of solving the delay-compensation problem with a *feedback* architecture. Such an architecture can, in principle, solve this problem, as shown earlier by Platt & Faggin [7]. Torkkola [9] also generalised the feedback architecture to remove dependencies across time, to achieve the deconvolution of mixtures which have been filtered, as in eq.1.

Here we propose a slightly different architecture than Torkkola's ([9], eq.15). His architecture could fail since it is missing feedback cross-weights for $t = 0$, ie: $w_{ij0}$. A full feedback system looks like:

$$u_i(t) = x_i - \sum_{j=1}^{N} \sum_{k=0}^{M-1} w_{ijk} u_j(t - d_{ij} - k). \tag{3}$$

and is illustrated in Fig.1. Because terms in $u_i(t)$ appear on both sides, we rewrite this in vector terms: $\mathbf{u}(t) = \mathbf{x}(t) - \mathbf{W}_0 \mathbf{u}(t) - \sum_{k=1}^{M-1} \mathbf{W}_k \mathbf{u}(t - k)$, in order to solve it as follows:

$$\mathbf{u}(t) = (\mathbf{I} + \mathbf{W}_0)^{-1} (\mathbf{x}(t) - \sum_{k=1}^{M-1} \mathbf{W}_k \mathbf{u}(t - k)) \tag{4}$$

In these equations, there is a feedback unmixing matrix, $\mathbf{W}_k$, for each time point of the filter, but the 'leading matrix', $\mathbf{W}_0$ has a special status in solving for $\mathbf{u}(t)$. The delay terms are useful since one metre of distance in air at an 8kHz sampling rate, corresponds to a whole 25 zero-taps of a filter. Reintroducing them gives us:

$$\mathbf{u}(t) = (\mathbf{I} + \mathbf{W}_0)^{-1} (\mathbf{x}(t) - \mathbf{net}(t)), \qquad net_i(t) = \sum_{j=1}^{N} \sum_{k=1}^{M-1} w_{ijk} u(t - d_{ij} - k)) \tag{5}$$

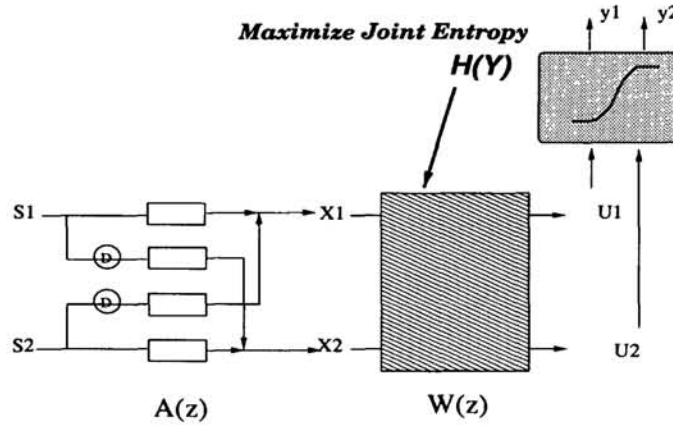

Figure 1: The feedback neural architecture of eq.9, which is used to separate and deconvolve signals. Each box represents a causal filter and each circle denotes a time delay.

## 3   Algorithms.

Learning in this architecture is performed by maximising the joint entropy, $H(\mathbf{y}(t))$, of the random vector $\mathbf{y}(t) = g(\mathbf{u}(t))$, where $g$ is a bounded monotonic nonlinear function (a sigmoid function). The success of this for separating sources depends on four assumptions: (1) that the sources are statistically independent, (2) that each source is white, ie: there are no dependencies between time points, (3) that the non-linearity, $g$, has a derivative which has higher kurtosis than the probability density functions (pdf's) of the sources, and (4) that a stable IIR (feedback) inverse of the mixing exists; ie: that $\mathbf{a}$ is *minimum phase* (see section 5).

Assumption (1) is reasonable and Assumption (3) allows some tailoring of our algorithm to fit data of different types. Assumption (2), on the other hand, is not true for natural signals. Our algorithm will whiten: it will remove dependencies across time which already existed in the original source signals, $s_i$. However, it is possible to restore the characteristic autocorrelations (amplitude spectra) of the sources by post-processing. For the reasoning behind Assumption (3) see [2]. We will discuss Assumption 4 in section 5.

In the static feedback case of eq.5, when $M = 1$, the learning rule for the feedback weights $\mathbf{W}_0$ is just a co-ordinate transform of the rule for feedforward weights, $\hat{\mathbf{W}}_0$, in the equivalent architecture of $\mathbf{u}(t) = \hat{\mathbf{W}}_0\mathbf{x}(t)$. Since $\hat{\mathbf{W}}_0 \equiv (\mathbf{I} + \mathbf{W}_0)^{-1}$, we have $\mathbf{W}_0 = \hat{\mathbf{W}}_0^{-1} - \mathbf{I}$, which, due to the quotient rule for matrix differentiation, differentiates as:

$$\Delta\mathbf{W}_0 = -(\hat{\mathbf{W}}^{-1})\Delta\hat{\mathbf{W}}(\hat{\mathbf{W}}^{-1}) \tag{6}$$

The best way to maximise entropy in the feedforward system is not to follow the entropy gradient, as in [2], but to follow its 'natural' gradient, as reported by Amari et al [1]:

$$\Delta\hat{\mathbf{W}} \propto \frac{\partial H(\mathbf{y})}{\partial \hat{\mathbf{W}}}\hat{\mathbf{W}}^T\hat{\mathbf{W}} \tag{7}$$

This is an optimal rescaling of the entropy gradient [1, 3]. It simplifies the learning

rule and speeds convergence considerably. Evaluated, it gives [2]:

$$\Delta \hat{\mathbf{W}}_0 \propto (\mathbf{I} + \hat{\mathbf{y}}\mathbf{u}^T)\hat{\mathbf{W}}_0, \qquad \hat{y}_i = \frac{\partial}{\partial y_i}\frac{\partial y_i}{\partial u_i} \tag{8}$$

Substituting into eq.7 gives the natural gradient rule for static feedback weights:

$$\Delta \mathbf{W}_0 \propto -(\mathbf{I} + \mathbf{W}_0)(\mathbf{I} + \hat{\mathbf{y}}\mathbf{u}^T), \tag{9}$$

This reasoning may be extended to networks involving filters. For the feedforward filter architecture $\mathbf{u}(t) = \sum_{k=0}^{M-1}\hat{\mathbf{W}}_k\mathbf{x}(t-k)$, we derive a natural gradient rule (for $k > 0$) of:

$$\Delta \hat{\mathbf{W}}_k \propto \hat{\mathbf{y}}\mathbf{u}_{t-k}^T\hat{\mathbf{W}}_k \tag{10}$$

where, for convenience, time has become subscripted. Performing the same coordinate transforms as for $\mathbf{W}_0$ above, gives the rule:

$$\Delta \mathbf{W}_k \propto -(\mathbf{I} + \mathbf{W}_k)\hat{\mathbf{y}}\mathbf{u}_{t-k}^T \tag{11}$$

(We note that learning rules similar to these have been independently derived by Cichocki et al [4]). Finally, for the delays in eq.5, we derive [2, 8]:

$$\Delta d_{ij} \propto \frac{\partial H(\mathbf{y})}{\partial d_{ij}} = -\hat{y}_i \sum_{k=1}^{M-1}\frac{\partial}{\partial t}w_{ijk}u(t - d_{ij} - k) \tag{12}$$

This rule is different from that in [8] because it uses the collected temporal gradient information from all the taps. The algorithms of eq.9, eq.11 and eq.12 are the ones we use in our experiments on the architecture of eq.5.

## 4  Simulation results for the feedback architecture

To test the learning rules in eq.9, eq.11 and eq.12 we used an IIR filter system to recover two sources which had been mixed and delayed as follows (in Z-transform notation):

$$\begin{aligned} A_{11}(z) &= 0.9 + 0.5z^{-1} + 0.3z^{-2} \\ A_{21}(z) &= -0.7z^{-5} - 0.3z^{-6} - 0.2z^{-7} \\ A_{12}(z) &= 0.5z^{-5} + 0.3z^{-6} + 0.2z^{-7} \\ A_{22}(z) &= 0.8 - 0.1z^{-1} \end{aligned} \tag{13}$$

The mixing system, $\mathbf{A}(z)$, is a minimum-phase system with all its zeros inside the unit circle. Hence, $\mathbf{A}(z)$ can be inverted using a stable causal IIR system since all poles of the inverting systems are also inside the unit circle. For this experiment, we chose an artificially-generated source: a white process with a Laplacian distribution $[f_x(x) = \exp(-|x|)]$. In the frequency domain the deconvolving system looks as follows:

$$\begin{bmatrix} U_1(z) \\ U_2(z) \end{bmatrix} = \frac{1}{D(z)}\begin{bmatrix} W_{11}(z) & W_{21}(z) \\ W_{12}(z) & W_{22}(z) \end{bmatrix}\begin{bmatrix} X_1(z) \\ X_2(z) \end{bmatrix} \tag{14}$$

where $D(z) = W_{11}(z)W_{22}(z) - W_{12}(z)W_{21}(z))$. This leads to the following solution for the weight filters:

$$\begin{aligned} W_{11}(z) &= A_{22}(z) & W_{22}(z) &= A_{11}(z) \\ W_{21}(z) &= -A_{21}(z) & W_{12}(z) &= -A_{12}(z) \end{aligned} \tag{15}$$

The learning rule we used was that of eq.9 and eq.11 with the logistic non-linearity, $y_i = 1/\exp(-u_i)$. Fig.2A shows the four filters learnt by our IIR algorithm. The bottom row shows the inverting system convolved with the mixing system, proving that $\mathbf{W} * \mathbf{A}$ is approximately the identity mapping. Delay learning is not demonstrated here, though for periodic signals like speech we observed that it is subject to local minima problems [8, 9].

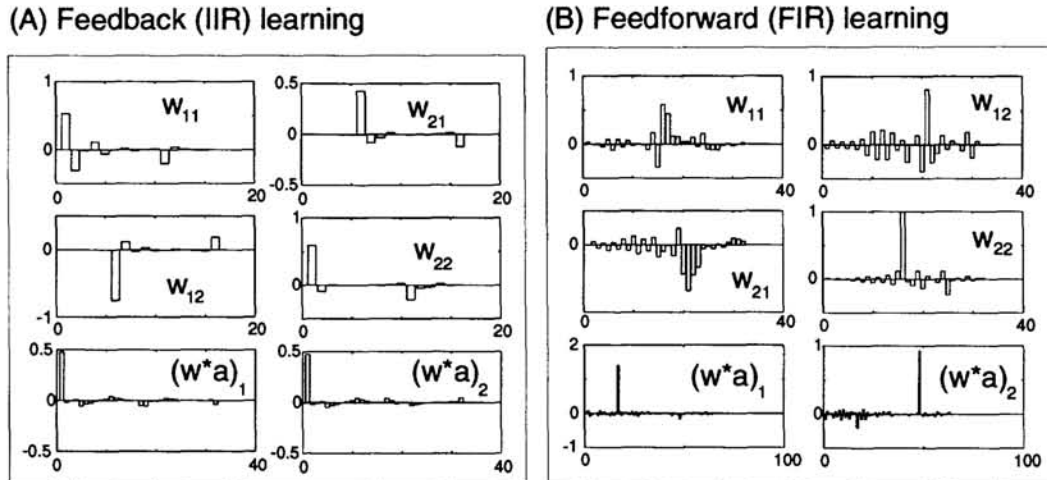

Figure 2: Top two rows: learned unmixing filters for (A) IIR learning on minimum-phase mixing, and (B) FIR freq.-domain learning on non-minimum phase mixing. Bottom row: the convolved mixing and unmixing systems. The delta-like response indicates successful blind unmixing. In (B) this occurs acausally with a time-shift.

## 5    Back to the feedforward architecture.

The feedback architecture is elegant but limited. It can only invert minimum-phase mixing (all zeros are inside the unit circle meaning that all poles of the inverting system are as well). Unfortunately, real room acoustics usually involves non-minimum phase mixing.

There does exist, however, a stable *non-causal* feedforward (FIR) inverse for non-minimum phase mixing systems. The learning rules for such a system can be formulated using the FIR polynomial matrix algebra as described by Lambert [5]. This may be performed in the time or frequency domain, the only requirements being that the inverting filters are long enough and their main energy occurs more-or-less in their centre. This allows for the non-causal expansion of the non-minimum phase roots, causing the roughly symmetrical "flanged" appearance of the filters in Fig.2B.

For convenience, we formulate the infomax and natural gradient infomax rules [2, 1] in the frequency domain:

$$\Delta \underline{\mathbf{W}} \propto \underline{\mathbf{W}}^{-H} + \mathrm{fft}(\hat{\mathbf{y}})\underline{\mathbf{X}}^H \tag{16}$$

$$\Delta \underline{\mathbf{W}} \propto (\mathbf{I} + \mathrm{fft}(\hat{\mathbf{y}})\underline{\mathbf{U}}^H)\underline{\mathbf{W}} \tag{17}$$

where the $H$ superscript denotes the Hermitian transpose (complex conjugate). In these rules, as in eq.14, $\underline{\mathbf{W}}$ is a matrix of filters and $\underline{\mathbf{U}}$ and $\underline{\mathbf{X}}$ are blocks of multi-

sensor signal in the frequency domain. Note that the nonlinearity $\hat{y}_i = \frac{\partial}{\partial y_i} \frac{\partial y_i}{\partial u_i}$ still operates in the time domain and the fft is applied at the output.

## 6  Simulation results for the feedforward architecture

To show the learning rule in eq.17 working, we altered the transfer function in eq.13 as follows:

$$A_{11}(z) = 1 + 1.0z^{-1} - 0.75z^{-2}. \tag{18}$$

This system is now non-minimum phase, having zeros outside the unit circle. The inverse system can be approximated by stable non-causal FIR filters. These were learnt using the learning rule of eq.17 (again, with the logistic non-linearity). The resulting learnt filters are shown in Fig.2B where the leading weights were chosen to be at half the filter size ($M/2$). Non-causality of the filters can be clearly observed for $w_{12}$ and $w_{21}$, where there are non-zero coefficients before the maximum amplitude weights. The bottom row of Fig.2B shows the successful separation by plotting the complete unmixing/mixing transfer function: $\mathbf{W} * \mathbf{A}$.

## 7  Experiments with real recordings

To demonstrate separation in a real room, we set up two microphones and recorded firstly two people speaking and then one person speaking with music in the background. The microphones and the sources were both 60cm apart and 60cm from each other (arranged in a square), and the sampling was 16kHz. Fig.3A shows the two recordings of a person saying the digits "one" to "ten" while loud music plays in the background. The IIR system of eq.5, eq.9 and eq.11 was unable to separate these signals, presumably due to the non-minimum-phase nature of the room transfer functions. However, the algorithm of eq.17, converged after 30 passes through the 10 second recordings. The filter lengths were 256 (corresponding to 16ms). The separated signals are shown in Fig.3B. Listening to them conveys a sense of almost-clean separation, though interference is audible. The results on the two people speaking were similar.

An important application is in spontaneous speech recognition tasks where the best recognizer may fail completely in the presence of background music or competing speakers (as in the teleconferencing problem). To test this application, we fed into a speech recognizer, ten sentences recorded with loud music in the background and ten sentences recorded with a simultaneous speaker interference. After separation, the recognition rate increased considerably for both cases. These results are reported in detail in [6].

## 8  Conclusions

Starting with 'Natural gradient infomax' IIR learning rules for blind time delay adjustment, separation and deconvolution, we showed how these worked well on minimum-phase mixing, but not on non-minimum-phase mixing, as usually occurs in rooms. This led us to an FIR frequency domain infomax approach suggested by Lambert [5]. The latter approach shows much better separation of speech and music mixed in a real-room. Based on these techniques, it should now be possible to develop real-world applications.

(A) Mixtures                                          (B) Separations

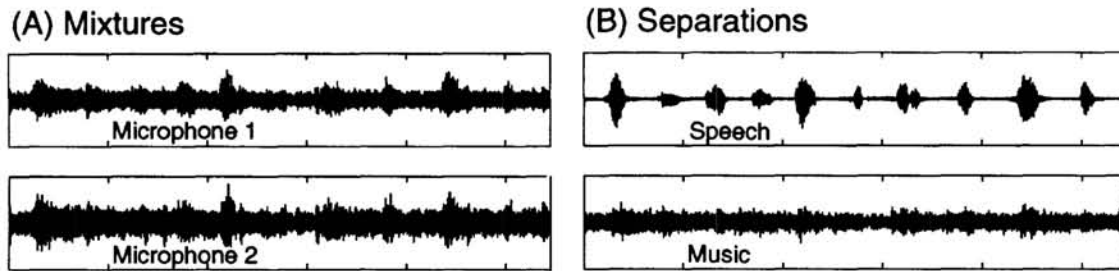

Figure 3: Real-room separation/deconvolution. (A) recorded mixtures (B) separated speech (spoken digits 1-10) and music.

## Acknowledgments

T.W.L. is supported by the Daimler-Benz-Fellowship, and A.J.B. by a grant from the Office of Naval Research. We are grateful to Kari Torkkola for sharing his results with us, and to Jürgen Fritsch, Terry Sejnowski and Alex Waibel for discussions and comments.

## References

[1] Amari S-I. Cichocki A. & Yang H.H. 1996. A new learning algorithm for blind signal separation, *Advances in Neural Information Processing Systems 8*, MIT press.

[2] Bell A.J. & Sejnowski T.J. 1995. An information maximisation approach to blind separation and blind deconvolution, *Neural Computation*, 7, 1129-1159

[3] Cardoso J-F. & Laheld B. 1996. Equivariant adaptive source separation, *IEEE Trans. on Signal Proc.*, Dec. 1996

[4] Cichocki A., Amari S-I & Cao J. 1996. Blind separation of delayed and convolved signals with self-adaptive learning rate, in *Proc. Intern. Symp. on Nonlinear Theory and Applications (NOLTA*96)*, Kochi, Japan.

[5] Lambert R. 1996.Multichannel blind deconvolution: FIR matrix algebra and separation of multipath mixtures, *PhD Thesis*, University of Southern California, Department of Electrical Engineering, May 1996.

[6] Lee T-W. & Orglmeister R. Blind source separation of real-world signals. submitted to *Proc. ICNN*, Houston, USA, 1997.

[7] Platt J.C. & Faggin F. 1992. Networks for the separation of sources that are superimposed and delayed, in Moody J.E et al (eds) *Advances in Neural Information Processing Systems 4*, Morgan-Kaufmann

[8] Torkkola K. 1996. Blind separation of delayed sources based on information maximisation, *Proc IEEE ICASSP*, Atlanta, May 1996.

[9] Torkkola K. 1996. Blind separation of convolved sources based on information maximisation, *Proc. IEEE Workshop on Neural Networks and Signal Processing*, Kyota, Japan, Sept. 1996